# Large-Scale Prediction of Disulphide Bond Connectivity

**Pierre Baldi    Jianlin Cheng**
Schoolof Information and Computer Science
University of California, Irvine
Irvine, CA 92697-3425
*{pfbaldi,jianlinc}@ics.uci.edu*

**Alessandro Vullo**
Computer Science Department
University College Dublin
Dublin, Ireland
*alessandro.vullo@ucd.ie*

## Abstract

The formation of disulphide bridges among cysteines is an important feature of protein structures. Here we develop new methods for the prediction of disulphide bond connectivity. We first build a large curated data set of proteins containing disulphide bridges and then use 2-Dimensional Recursive Neural Networks to predict bonding probabilities between cysteine pairs. These probabilities in turn lead to a weighted graph matching problem that can be addressed efficiently. We show how the method consistently achieves better results than previous approaches on the same validation data. In addition, the method can easily cope with chains with arbitrary numbers of bonded cysteines. Therefore, it overcomes one of the major limitations of previous approaches restricting predictions to chains containing no more than 10 oxidized cysteines. The method can be applied both to situations where the bonded state of each cysteine is known or *unknown*, in which case bonded state can be predicted with 85% precision and 90% recall. The method also yields an estimate for the total number of disulphide bridges in each chain.

## 1   Introduction

The formation of covalent links among cysteine (Cys) residues with disulphide bridges is an important and unique feature of protein folding and structure. Simulations [1], experiments in protein engineering [15, 8, 14], theoretical studies [7, 18], and even evolutionary models [9] stress the importance of disulphide bonds in stabilizing the native state of proteins. Disulphide bridges may link distant portions of a protein sequence, providing strong structural constraints in the form of long-range interactions. Thus prediction/knowledge of the disulphide connectivity of a protein is important and provides essential insights into its structure and possibly also into its function and evolution.

Only recently has the problem of predicting disulphide bridges received increased attention. In the current literature, this problems is typically split into three subproblems: (1) prediction of whether a protein chain contains intra-chain disulphide bridges or not; (2) prediction of the intra-chain bonded/non-bonded state of individual cysteines; and (3) prediction of intra-chain disulphide bridges, i.e. of the actual pairings between bonded cysteines (see Fig.1). In this paper, we address the problem of intra-chain connectivity prediction, and

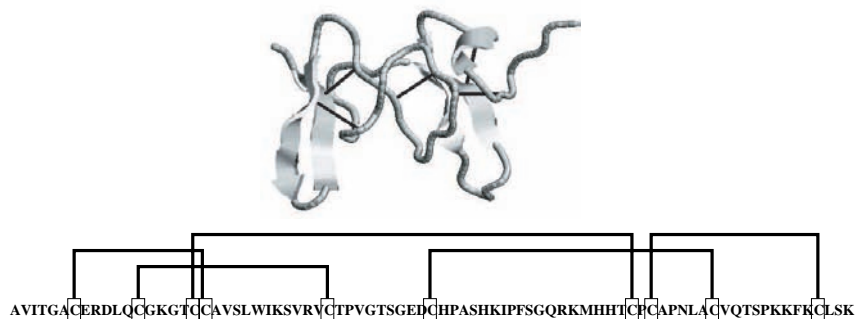

Figure 1: Structure (top) and connectivity pattern (bottom) of intestinal toxin 1, PDB code 1IMT. Disulphide bonds in the structure are shown as thick lines.

specifically the solution of problem (3) alone, and of problems (2) and (3) simultaneously.

Existing approaches to connectivity prediction use stochastic global optimization [10], combinatorial optimization [13] and machine learning techniques [11, 17]. The method in [10] represents the set of potential disulphide bridges in a sequence as a complete weighted undirected graph. Vertices are oxidized cysteines and edges are labeled by the strength of interaction (contact potential) in the associated pair of cysteines. A simulated annealing approach is first used to find an optimal set of weights. After a complete labeled graph is obtained, candidate bridges are then located by finding the maximum weight perfect matching[1].

The method in [17] attempts to solve the problem using a different machine learning approach. Candidate connectivity patterns are modelled as undirected graphs. A recursive neural network architecture is trained to score candidate patterns according to a similarity metric with respect to correct graphs. Vertices of the graphs are labeled by fixed-size vectors corresponding to multiple alignment profiles in a local window around each cysteine. During prediction, the score computed by the network is used to exhaustively search the space of candidate graphs. This method, tested on the same data as in [11], achieved the best results. Unfortunately, for computational reasons, both this method and the previous one can only deal with sequences containing a limited number of bonds ($K \leq 5$).

A different approach to predicting disulphide bridges is reported in [13], where finding disulphide bridges is part of a more general protocol aimed at predicting the topology of $\beta$-sheets in proteins. Residue-to-residue contacts (including Cys-Cys bridges) are predicted by solving a series of integer linear programming problems in which customized hydrophobic contact energies must be maximized. This method cannot be compared with the other approaches because the authors report validation results only for two relatively short polypeptides with few bonds (2 and 3).

In this paper we use 2-Dimensional Recursive Neural Network (2D-RNN, [4]) to predict disulphide connectivity in proteins starting from their primary sequence and its homologues. The output of 2D-RNN are the pairwise probabilities of the existence of a bridge between any pair of cysteines. Candidate disulphide connectivities are predicted by finding the maximum weight perfect matching. The proposed framework represents a significant improvement in disulphide connectivity prediction for several reasons. First, we show how the method consistently achieves better results than all previous approaches on the same validation data. Second, our architecture can easily cope with chains with arbitrary number

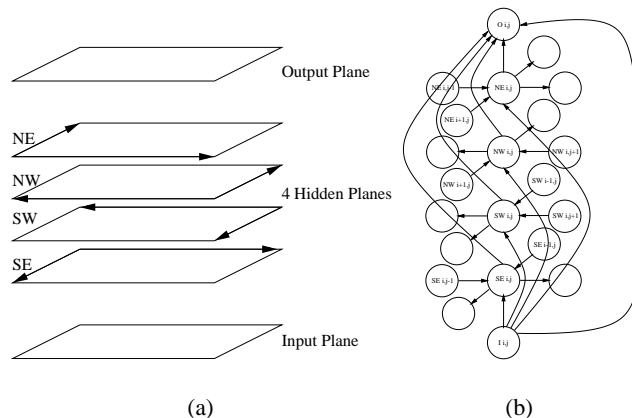

(a)            (b)

Figure 2: (a) General layout of a 2D-RNN for processing two-dimensional objects such as disulphide contacts, with nodes regularly arranged in one input plane, one output plane, and four hidden planes. In each plane, nodes are arranged on a square lattice. The hidden planes contain directed edges associated with the square lattices. All the edges of the square lattice in each hidden plane are oriented in the direction of one of the four possible cardinal corners: NE, NW, SW, SE. Additional directed edges run vertically in column from the input plane to each hidden plane, and from each hidden plane to the output plane. (b) Connections within a vertical column $(i, j)$ of the directed graph. $I_{ij}$ represents the input, $O_{ij}$ the output, and $NE_{ij}$ represents the hidden variable in the North-East hidden plane.

of bonded cysteines. Therefore, it overcomes the limitation of previous approaches which restrict predictions to chains with no more than 10 oxidized cysteines. Third, our methods can be applied both to situations where the bonded state of each cysteine is known or unknown. And finally, once trained, our system is very rapid and can be used on a high-throughput scale.

## 2   Methods

**Algorithms**

To predict disulphide connectivity patterns, we use the 2D-RNN approach described in [4], whereby a suitable Bayesian network is recast, for computational effectiveness, in terms of recursive neural networks, where local conditional probability tables in the underlying directed graph are replaced by deterministic relationships between a variable and its parent node variables. These functions are parameterized by neural networks using appropriate weight sharing as described below. Here the underlying directed graph for disulphide connectivity has six 2D-layers: input, output, and four hidden layers (Figure 2(a)). Vertical connections, within an $(i, j)$ column, run from input to hidden and output layers, and from hidden layers to output (Figure 2(b)). In each one of the four hidden planes, square lattice connections are oriented towards one of the four cardinal corners. Detailed motivation for these architectures can be found in [4] and a mathematical analysis of their relationships to Bayesian networks in [5]. The essential point is that they combine the flexibility of graphical models with the deterministic propagation and learning speed of artificial neural networks. Unlike traditional neural networks with fixed-size input, these architectures can process inputs of variable structure and length, and allow lateral propagation of contextual information over considerable length scales.

In a disulphide contact map prediction, the $(i, j)$ output represents the probability of whether the $i$-th and $j$-th cysteines in the sequence are linked by a disulphide bridge

or not. This prediction depends directly on the $(i, j)$ input and the four-hidden units in the same column, associated with omni-directional contextual propagation in the hidden planes. Hence, using weight sharing across different columns, the model can be summarized by 5 distinct neural networks in the form

$$
\begin{cases}
O_{ij} = \mathcal{N}_O(I_{ij}, H_{i,j}^{NW}, H_{i,j}^{NE}, H_{i,j}^{SW}, H_{i,j}^{SE}) \\
H_{i,j}^{NE} = \mathcal{N}_{NE}(I_{i,j}, H_{i-1,j}^{NE}, H_{i,j-1}^{NE}) \\
H_{i,j}^{NW} = \mathcal{N}_{NW}(I_{i,j}, H_{i+1,j}^{NW}, H_{i,j-1}^{NW}) \\
H_{i,j}^{SW} = \mathcal{N}_{SW}(I_{i,j}, H_{i+1,j}^{SW}, H_{i,j+1}^{SW}) \\
H_{i,j}^{SE} = \mathcal{N}_{SE}(I_{i,j}, H_{i-1,j}^{SE}, H_{i,j+1}^{SE})
\end{cases}
\tag{1}
$$

where $\mathcal{N}$ represents NN parameterization. Learning can proceed by gradient descent (backpropagation) due to the acyclic nature of the underlying graph.

The input information is based on the sequence itself or rather the corresponding profile derived by multiple alignment methods to leverage evolutionary information, possibly augmented with secondary structure and solvent accessibility information derived from the PDB files and/or our SCRATCH suite of predictors [16, 3, 4]. For a sequence of length $N$ and containing $M$ cysteines, the output layer contains $M \times M$ units. The input and hidden layer can scale like $N \times N$ if the full sequence is used, or like $M \times M$ if only fixed-size windows around each cysteine are used, as in the experiments reported here. The results reported here are obtained using local windows of size 5 around each cysteine, as in [17]. The input of each position within a window is the normalized frequency of all 20 amino acids at that position in the multiple alignment generated by aligning the sequence with the sequences in the NR database using the PSI-BLAST program as described, for instance, in [16]. Gaps are treated as one additional amino acid. For each $(i, j)$ location an extra input is added to represent the absolute linear distance between the two corresponding cysteines.

Finally, it is essential to remark that the same 2D-RNN approach can be trained and applied here in two different modes. In the first mode, we can assume that the bonded state of the individual cysteines is known, for instance through the use of a specialized predictor for bonded/non-bonded states. Then if the sequence contains $M$ cysteines, $2K$ ($2K \leq M$) of which are intra-chain disulphide bonded, the prediction of the connectivity can focus on the $2K$ bonded cysteines exclusively and ignore the remaining $M - 2K$ cysteines that are not bonded. In the second mode, we can try to solve both prediction problems–bond state and connectivity–at the same time by focusing on all cysteines in a given sequence. In both cases, the output is an array of pairwise probabilities from which the connectivity pattern graph must be inferred. In the first case, the total number of bonds or edges in the connectivity graph is known ($K$). In the second case, the total number of edges must be inferred. In section 3, we show that sum of all probabilities across the output array can be used to estimate the number of disulphide contacts.

**Data Preparation**

In order to assess our method, two data sets of known disulphide connectivities were compiled from the Swiss-Prot archive [2]. First, we considered the same selection of sequences as adopted in [11, 17] and taken from the Swiss-Prot database release no. 39 (October 2000). Additionally, we collected and filtered a more recent selection of chains extracted from the latest available Swiss-Prot archive, version 41.19 (August 2003). In the following, we refer to these two data sets as SP39 and SP41, respectively.

SP41 was compiled with the same filtering procedure used for SP39. Specifically, only chains whose structure is deposited in the Protein Data Bank PDB [6] were retained. We filtered out proteins with disulphide bonds assigned tentatively or disulphide bonds inferred by similarity. We finally ended up with 966 chains, each with a number of disulphide bonds in the range of 1 to 24. As previously pointed out, our methodology is not limited by the

number of disulphide bonds, hence we were able to retain and test the algorithm on the whole filtered set of non-trivial chains. This set consists of 712 sequences, each containing at least two bridges ($K \geq 2$)–the case $K = 1$ being trivial when the bonded state is known. By comparison, SP39 includes 446 chains with no more than 5 bridges; SP41 additionally includes 266 sequences and 112 of these have more than 10 oxidized cysteines.

In order to avoid biases during the assessment procedure and to perform $k$-fold cross validation, SP41 was partitioned in ten different subsets, with the constraint that sequence similarity between two different subsets be less or equal to $30\%$. This is similar to the criteria adopted in [17, 10], where SP39 was splitted into four subsets.

**Graph Matching to Derive Connectivity from Output Probabilities**
In the case where the bonded state of the cysteines is known, one has a graph with $2K$ nodes, one for each bonded cysteine. The weight associated with each edge is the probability that the corresponding bridge exists, as computed by the predictor. The problem is then to find a connectivity pattern with $K$ edges and maximum weight, where each cysteine is paired uniquely with another cysteine. The maximum weight matching algorithm of Gabow [12] is used to chosen paired cysteines (edges), whose time complexity is cubic $O(V^3) = O(K^3)$, where $V$ is the number of vertices and linear $O(V) = O(K)$ space complexity beyond the storage of the graph. Note that because the number of bonded cysteines in general is not very large, it is also possible in many cases to use an exhaustive search of all possible combinations. Indeed, the number of combinations is $1 \times 3 \times 5 \times \ldots (2K-1)$ which yields 945 connectivity patterns in the case of 10 bonded cysteines.

The case where the bonded state of the cysteines is not known is slightly more involved and the Gabow algorithm cannot be applied directly since the graph has $M$ nodes but, if some of the cysteines are not bonded, only a subset of $2K < M$ nodes participate in the final maximum weighted matching. Alternatively, we use a greedy algorithm to derive the connectivity pattern using the estimate of the total number of bonds. First, we order the edges in decreasing order of probabilities. Then we pick the edge with the highest probability. Then we pick the next edge with highest probability that is not incident to the first edge and so forth, until $K$ edges have been selected. Because this greedy procedure is not guaranteed to find the global optimum, we find it useful to make it a little more robust by repeating $L$ times. In each run $i = 1, \ldots, L$, the first edge selected is the $i$-th most probable edge. In other words the different runs differ by the choice of the first edge, noting that in practice the optimal solution always contain one of the top $L$ edges. This procedure works well in practice because the edges with largest probabilities tend to occur in the final pattern. For $L$ reasonably large, the optimal connectivity pattern can usually be found. We have compared this method with Gabow's algorithm in the case where the bonding state is known and observed that when $L = 6$, this greedy heuristic yields results that are as good as those obtained with Gabow's algorithm which, in this case, is guaranteed to find a global optimum. The results reported here are obtained using the greedy procedure with $L = 6$. The advantage of the greedy algorithm is its low $O(LM^2)$ complexity time. It is important to note that this method ends up by producing a prediction of both the connectivity pattern and of the bonding state of each cysteine.

## 3   Results

**Disulphide Connectivity Prediction for Bonded Cysteines**
Here we assume that the bonding state is known. We train 2D-RNN architectures using the SP39 data set to compare with other published results. We evaluate the performance using the precision $P$ ($P$=TP/(TP+FP) with TP = true positives and FP = false positives) and recall $R$ ($R$=TP/(TP+FN) with FN = false negatives).

As shown in Table 1, in all but one case the results are superior to what has been previ-

| $K$ | Pair Precision | Pattern Precision |
|---|---|---|
| 2 | 0.74* (0.73) | 0.74* (0.73) |
| 3 | 0.61* (0.51) | 0.51* (0.41) |
| 4 | 0.44* (0.37) | 0.27* (0.24) |
| 5 | 0.41* (0.30) | 0.11  (0.13) |
| $2\ldots5$ | 0.56* (0.49) | 0.49* (0.44) |

Table 1: Disulphide connectivity prediction with 2D-RNN assuming the bonding state is known. Last row reports performance on all test chains. * denote levels of precision that exceeds previously reported best results in the literature [17] (in parentheses).

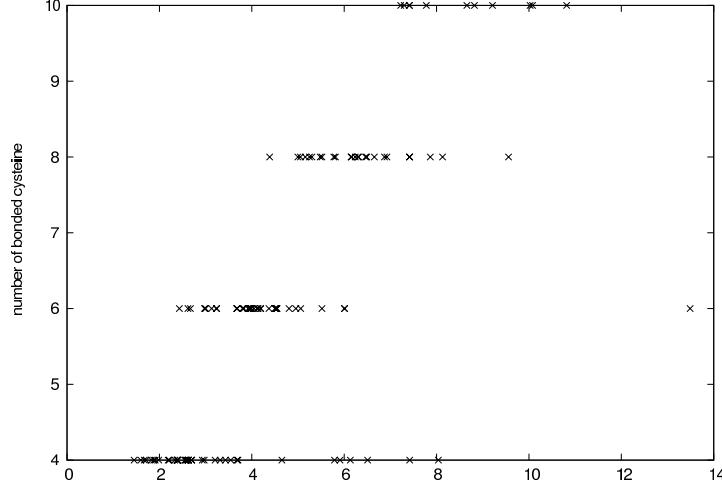

Figure 3: Correlation between number of bonded cysteines ($2K$) and $\sqrt{\sum_{i\neq j} O_{i,j}}\log M$.

ously reported in the literature [17, 11]. In some cases, results are substantially better. For instance, in the case of 3 bonded cysteines, the precision reaches 0.61 and 0.51 at the pair and pattern levels, whereas the best similar results reported in the literature are 0.51 (pair) and 0.41 (pattern).

**Estimation of the Number $K$ of Bonds**

Analysis of the prediction results shows that there is a relationship between the sum of all the probabilities, $\sum_{i\neq j} O_{i,j}$, in the graph (or the output layer of the 2D-RNN) and the total number of bonded cysteines ($2K$). For instance, on one of the cross-validation training sets, the correlation coefficient between $2K$ and $\sum_{i\neq j} O_{i,j}$ is 0.7, the correlation coefficient between $2K$ and $M$ is 0.68, and the correlation coefficient between $2K$ and $\sqrt{\sum_{i\neq j} O_{i,j}}\log M$ is 0.72. As shown in Figure 3, there is a reasonably linear relationship between the total number $2K$ of bonded cysteines and the product $\sqrt{\sum_{i\neq j} O_{i,j}}\log M$, where $M$ is the total number of cysteines in the sequence being considered. The slope and y-intercept for the line are respectively 0.66 and 3.01 on one training data set. Using this, we estimate the total number of bonded cysteines using linear regression and rounding off, making sure that the total number is even and does not exceed the total number of cysteines in the sequence. In the following experiments, the regression equation for predicting $K$ is solved separately based on each cross-validation training set.

| $K$ | Pair Recall | Pair Precision | Pattern Precision |
|---|---|---|---|
| 2 | 0.59 | 0.49 | 0.40 |
| 3 | 0.50 | 0.45 | 0.32 |
| 4 | 0.36 | 0.37 | 0.15 |
| 5 | 0.28 | 0.31 | 0.03 |

Table 2: Prediction of disulphide connectivity pattern with 2D-RNN on all the cysteines, without assuming knowledge of the bonding state.

**Disulphide Connectivity Prediction from Scratch**

In the last set of experiments, we do not assume any knowledge of the bonding state and apply the 2D-RNN approach to *all* the cysteines (both bonded and not bonded) in each sequence. We predict the number of bonds, the bonding state, and connectivity pattern using one predictor. Experiments are run both on SP39 (4-fold cross validation) and SP41 (10-fold cross validation).

For lack of space, we cannot report all the results but, for example, precision and recall for SP39 are given in Table 2 for $2 \leq K \leq 5$. Table 3 shows the kind of results that are obtained when the method is applied to sequences with more than $K = 5$ bonds in SP41. The pair precision remains quite good, although the results can be noisy for certain values because there are not many such examples in the data. Finally, the precision of bonded state prediction is 0.85, and the recall of bonded state prediction is 0.9. The precision and recall of bond number prediction is 0.68. The average absolute difference between true bond and predicted bond number is 0.42. The average absolute difference between true bond number and wrongly predicted bond number is 1.3.

| $K$ | 6 | 7 | 8 | 9 | 10 | 11 | 12 | 15 | 16 | 17 | 18 | 19 | 24 |
|---|---|---|---|---|---|---|---|---|---|---|---|---|---|
| Precision | 0.41 | 0.40 | 0.34 | 0.37 | 0.5 | 0.4 | 0.17 | 0.37 | 0.57 | 0.40 | 0.56 | 0.42 | 0.24 |

Table 3: Prediction of disulphide connectivity pattern with 2D-RNN on all the cysteines, without assuming knowledge of the bonding state and when the number of bridges $K$ exceeds 5.

## 4 Conclusion

We have presented a complete system for disulphide connectivity prediction in cysteine-rich proteins. Assuming knowledge of cysteine bonding state, the method outperforms existing approaches on the same validation data. The results also show that the 2D-RNN method achieves good recall and accuracy on the prediction of connectivity pattern even when the bonding state of individual cysteines is not known. Differently from previous approaches, our method can be applied to chains with $K > 5$ bonds and yields good, cooperative, predictions of the total number of bonds, as well as of the bonding states and bond locations. Training can take days but once trained predictions can be carried on a proteomic or protein engineering scale. Several improvements are currently in progress including (a) developing a classifier to discriminate protein chains that do not contain any disulphide bridges, using kernel methods; (b) assessing the effect on prediction of additional input information, such as secondary structure and solvent accessibility; (c) leveraging the predicted cysteine contacts in 3D protein structure prediction; and (d) curating a new larger training set. The current version of our disulphide prediction server DIpro (which includes step (a)) is available through: `http://www.igb.uci.edu/servers/psss.html`.

**Acknowledgments**

Work supported by an NIH grant, an NSF MRI grant, a grant from the University of California Systemwide Biotechnology Research and Education Program, and by the Institute for Genomics and Bioinformatics at UCI.

## Footnotes

[1]A perfect matching of a graph $(V, E)$ is a subset $E' \subseteq E$ such that each vertex $v \in V$ is met by only one edge in $E'$.

# References

[1] V.I. Abkevich and E.I. Shankhnovich. What can disulfide bonds tell us about protein energetics, function and folding: simulations and bioinformatics analysis. *J. Math. Biol.*, 300:975–985, 2000.

[2] A. Bairoch and R. Apweiler. The SWISS-PROT protein sequence database and its supplement TrEMBL. *Nucleic Acids Res.*, 28:45–48, 2000.

[3] P. Baldi and G. Pollastri. Machine learning structural and functional proteomics. *IEEE Intelligent Systems. Special Issue on Intelligent Systems in Biology*, 17(2), 2002.

[4] P. Baldi and G. Pollastri. The principled design of large-scale recursive neural network architectures–dag-rnns and the protein structure prediction problem. *Journal of Machine Learning Research*, 4:575–602, 2003.

[5] P. Baldi and M. Rosen-Zvi. On the relationship between deterministic and probabilistic directed graphical models. 2004. Submitted.

[6] H. M. Berman, J. Westbrook, Z. Feng, G. Gilliland, T. N. Bhat, H. Weissig, I. N. Shindyalov, and P. E. Bourne. The Protein Data Bank. *Nucl. Acids Res.*, 28:235–242, 2000.

[7] S. Betz. Disulfide bonds and the stability of globular proteins. *Proteins, Struct., Function Genet.*, 21:167–195, 1993.

[8] J. Clarke and A.R. Fersht. Engineered disulfide bonds as probes of the folding pathway of barnase - increasing stability of proteins against the rate of denaturation. *Biochemistry*, 32:4322–4329, 1993.

[9] L. Demetrius. Thermodynamics and kinetics of protein folding: an evolutionary perpective. *J. Theor. Biol.*, 217:397–411, 2000.

[10] P. Fariselli and R. Casadio. Prediction of disulfide connectivity in proteins. *Bioinformatics*, 17:957–964, 2001.

[11] P. Fariselli, P. L. Martelli, and R. Casadio. A neural network-based method for predicting the disulfide connectivity in proteins. In E. Damiani et al., editors, *Knowledge based intelligent information engineering systems and allied technologies (KES 2002)*, volume 1, pages 464–468. IOS Press, 2002.

[12] H.N. Gabow. An efficient implementation of Edmond's algorithm for maximum weight matching on graphs. *Journal of the ACM*, 23(2):221–234, 1976.

[13] J.L. Klepeis and C.A. Floudas. Prediction of $\beta$-sheet topology and disulfide bridges in polypeptides. *J. Comput. Chem.*, 24:191–208, 2003.

[14] T.A. Klink, K.J. Woycechosky, K.M. Taylor, and R.T. Raines. Contribution of disulfide bonds to the conformational stability and catalytic activity of ribonuclease A. *Eur. J. Biochem.*, 267:566–572, 2000.

[15] M. Matsumura et al. Substantial increase of protein stability by multiple disulfide bonds. *Nature*, 342:291–293, 1989.

[16] G. Pollastri, D. Przybylski, B. Rost, and P. Baldi. Improving the prediction of protein secondary structure in three and eight classes using recurrent neural networks and profiles. *Proteins*, 47:228–235, 2002.

[17] A. Vullo and P. Frasconi. Disulfide connectivity prediction using recursive neural networks and evolutionary information. *Bioinformatics*, 20:653–659, 2004.

[18] W.J. Wedemeyer, E. Welkler, M. Narayan, and H.A. Scheraga. Disulfide bonds and protein-folding. *Biochemistry*, 39:4207–4216, 2000.
